# KODAK IMAGELINK™ OCR
# Alphanumeric Handprint Module

**Alexander Shustorovich and Christopher W. Thrasher**
Business Imaging Systems, Eastman Kodak Company, Rochester, NY 14653-5424

## ABSTRACT

This paper describes the Kodak Imagelink ™ OCR alphanumeric handprint module. There are two neural network algorithms at its core: the first network is trained to find individual characters in an alphanumeric field, while the second one performs the classification. Both networks were trained on Gabor projections of the original pixel images, which resulted in higher recognition rates and greater noise immunity. Compared to its purely numeric counterpart (Shustorovich and Thrasher, 1995), this version of the system has a significant application specific postprocessing module. The system has been implemented in specialized parallel hardware, which allows it to run at 80 char/sec/board. It has been installed at the Driver and Vehicle Licensing Agency (DVLA) in the United Kingdom, and its overall success rate exceeds 96% (character level without rejects), which translates into 85% field rate. If approximately 20% of the fields are rejected, the system achieves 99.8% character and 99.5% field success rate.

## 1  INTRODUCTION

The system we describe below was designed to process alphanumeric fields extracted from forms. The major assumptions were that (1) the form layout and definition allows the system to capture the field image with a single line of characters, (2) the characters are handprinted capital letters and numerals, with possible addition of several special characters, and (3) the characters may occasionally touch, but generally they do not overlap. We also assume that some additional information about the contents of the field is available to assist in the process of disambiguation. Otherwise, it is virtually impossible to distinguish not only between " O " and zero, but also " I " and one, " Z " and two, " S " and five, etc.

A good example of such an application is the processing of vehicle registration forms at the Driver and Vehicle Licensing Agency (DVLA) in the United Kingdom. The alphanumeric field in question contains a license plate. There are 29 allowed patterns of character combinations, from two to seven characters long. For example, " A999AAA " is a valid license, whereas " A9A9A9 " is not (here " A " stands for any alpha character, " 9 " - for any numeric character). In addition, every field has a

control character box on the right. This control character is computed as a remainder of the integer division by 37 of a linear combination of numeric values of the characters in the main field. Ambiguous characters, namely " O ", " I ", and " S " are not allowed in the role of the control character, so they are replaced here by " - ", " + ", and " / " (not a very good choice, and the 37th character used is the " % ". To make things more complicated, sometimes the control character is not available at the moment of filing the form (at a local post office), and this lack of knowledge is indicated by putting an asterisk instead. Later we will discuss possible ways to use this additional information in an application specific postprocessing module.

## 2  SEGMENTATION AND ALTERNATIVE APPROACHES

The most challenging problem for handprint OCR is finding individual characters in a field. A number of approaches to this problem can be found in the literature, the two most common being (1) segmentation (Gupta et al., 1993, as an example of a recent publication), and (2) combined segmentation and recognition (Keeler and Rumelhart, 1992).

The segmentation approach has difficulty separating touching characters, and recently the consensus of practitioners in the field started shifting towards combined segmentation and recognition. In this scheme, the algorithm moves a window of a certain width along the field, and confidence values of competing classification hypotheses are used (sometimes with a separate centered/noncentered node) to decide if the window is positioned on top of a character. In the Saccade system (Martin et al., 1993), for example, the neural network was trained not only to recognize characters in the center of the moving window (and whether there is a character centered in the window), but also to make corrective jumps (saccades) to the nearest character and, after classification, to the next character.

Still another variation on the theme is an arrangement when the classification window is duplicated with one- or several-pixel shifts along the field (Benjio et al., 1994). Then the outputs of the classifiers serve as input for a postprocessing module (in this paper, a Hidden Markov Model) used to decide which of the multitude of processing windows actually have centered characters in them.

All these approaches have deficiencies. As we mentioned earlier, touching characters are difficult for autonomous segmenters. The moving (and jumping) window with a single centered/noncentered node tends to miss narrow characters and sometimes to duplicate wide ones. The replication of a classifier together with postprocessing tends to be quite expensive computationally.

## 3  POSITIONING NETWORK

To do the positioning, we decided to introduce an array of output units corresponding to successive pixels in the middle portion of the window. These nodes signal if a center ("heart") of a character lies at the corresponding positions. Because the precision with which a human operator can mark the character heart is low (usually within one or two pixels at best), the target activations of three consecutive nodes are set to one if there is a character heart at a pixel position corresponding to the middle node. The rest of the target activations are set to zero.

The network is then trained to produce bumps of activation indicating the character hearts. Two buffer regions on the left and on the right of the window (pixels without corresponding output nodes) are necessary to allow all or most of the character centered at each of the output node positions to fit inside the window. The replacement of a single centered/noncentered node by an array allows us to average output activations, generated by different window shifts, while corresponding to the same position. This additional procedure allows us to slide the window several pixels

at a time: the appropriate step is a trade-off between the processing speed and the required level of robustness. The final procedure involves thresholding of the activation-wave and the estimation of the predicted character position as the center of mass of the activation-bubble. The resulting algorithm is very effective: touching characters do not present significant problems, and only abnormally wide characters sometimes fool the system into false alarms.

The system works with preprocessed images. Each field is divided into subfields of disconnected groups of characters. These subfields are size-normalized to a height of 20 pixels. After that they are reassembled into a single field again, with 6 pixel gaps between them. Two blank rows are added both along the top and the bottom of the recombined field as preferred by the Gabor projection technique (Shustorovich, 1994). In our current system, the input nodes of a sliding window are organized in a 24 x 36 array. The first, intermediary, layer of the network implements the Gabor projections. It has 12 x 12 local receptive fields (LRFs) with fixed precomputed weights. The step between LRFs is 6 pixels in both directions. We work with 16 Gabor basis functions with circular Gaussian envelopes centered within each LRF; they are both sine and cosine wavelets in four orientations and two sizes. All 16 projections from each LRF constitute the input to a column of 20 hidden units, thus the second (first trainable) hidden layer is organized in a three-dimensional array 3 x 5 x 20. The third hidden layer of the network also has local receptive fields, they are three-dimensional 2 x 2 x 20 with the step 1 x 1 x 0. The units in the third hidden layer are also duplicated 20 times, thus this layer is organized in a three-dimensional array 2 x 4 x 20. The fourth hidden layer has 60 units fully connected to the third layer. Finally, the output layer has 12 units, also fully connected to the fourth layer.

The network was trained using a variant of the Back-Propagation algorithm. Both training and testing sets were drawn from the field data collected at DVLA. The training set contained approximately 60,000 characters from 8,000 fields, and about 5,000 characters from 650 fields were used for testing. On this test set, more than 92% of all character hearts were found within 1-pixel precision, and only 0.4% were missed by more than 4 pixels.

## 4   CLASSIFICATION NETWORK

The structure of the classification network resembles that of the positioning network. The Gabor projection layer works in exactly the same way, but the window size is smaller, only 24 x 24 pixels. We chose this size because after height normalization to 20 pixels, only occasionally the characters are wider than 24 pixels. Widening the window complicates training: it increases the dimensionality of the input while providing information, mostly about irrelevant pieces of adjacent characters. As a result, the second layer is organized as a 3 x 3 x 20 array of units with LRFs and shared weights, the third is a 2 x 2 x 20 array of units with LRFs, and there are 37 output units fully connected to the 80 units in the third layer. The number of output units in this variant of our system has been determined by the intended application. It was necessary to recognize uppercase letters, numerals, and also five special characters, namely plus (+), minus (-), slash (/), percent (%), and asterisk (*). Since additional information was available for the purposes of disambiguation, we combined " O " and zero, " I " and one, " Z " and two, " S " and five, and so the number of output classes became 26 (alpha) + 6 (numerals 3,4,6,7,8,9) + 5 (special characters) = 37.

Because we did not expect any positioning module to provide precision higher than 1 or 2 pixels, the classifier network was trained and tested on five copies of all centered characters in the database, with shifts of 0, 1, and 2 pixels, both left and right. On the same test set mentioned in the previous section, the corresponding character recognition rates averaged 93.0%, 95.5%, and 96.0% for characters normalized to the

height of 18 to 20 pixels and placed in the middle of the window with shifts of 0 and 1 pixel up and down.

## 5 POSTPROCESSING MODULE

The postprocessing module is a rule-based algorithm. First, it monitors the width of each subfield and rejects it if the number of predicted character hearts is inconsistent with the width. For example, if the positioning system cannot find a single character in a subfield, the output of the system becomes a question mark. Second, the postprocessing module organizes competition between predicted character hearts if they are too close to each other. For example, it will kill a predicted center with a lower activation value if its distance from a competitor is less than ten pixels, but it may allow both to survive if one of the two labels is "one". It is especially sensitive to closely positioned centers with identical labels, and will remove the weaker one for wide characters such as " W " or " M ".

The rest of the postprocessing had to rely on the application knowledge. Since the alphanumeric fields on DVLA forms contain license plates, we could use the fact that there are exactly 29 allowed patterns of symbol combinations, and that correct strings should match control characters from the box on the right.

Because in this application rejection of individual characters is meaningless, we decided to keep and analyze all possible candidates for each detected position, that is, characters with output activations above a certain threshold (currently, 0.1). Of course, special characters are not allowed in the main field. The field as a whole is rejected if for any one position there is not even a single candidate character. All possible combinations of candidate characters are analyzed. A candidate string is rejected if it does not conform to any of allowed patterns, or if it does not match any of the candidate control characters. All remaining candidate strings are assigned confidences. Since a chain is no stronger than its weakest link, in the case of an asterisk (no control character information), the string confidence equals that of its least confident character. If there is a valid control character, then we can tolerate one low-confidence character, and so the string confidence equals that of its character with the second lowest individual confidence. If there are two or more candidate strings, the difference in confidence between the best and the second best is compared to another threshold (currently, 0.7) in order to pass the final round of rejects.

## 6 CONCLUSIONS

Kodak Imagelink™ OCR alphanumeric handprint module described in this paper uses one neural network to find individual characters in a field, and then the second network performs the classification. The outputs of both networks are interpreted by a postprocessing module that generates the final label string (Figure 1, Figure 2).

The algorithms were designed within the constraints of the planned hardware implementation. At the same time, they provide a high level of positioning accuracy as well as classification ability. One new feature of our approach is the use of an array of centered/noncentered nodes to significantly improve speed and robustness of the positioning scheme. The overall robustness of the system is further improved by noise resistance provided by a layer of Gabor projection units. The positioning module and the classification module are unified by the postprocessing module.

System-level testing was performed on a test set mentioned above. The image quality was generally very good, but the data included some fields with touching characters. The character level success rate (without rejects) achieved on this test exceeded 96%, which corresponded to above 85% field rate. With approximately 20% of the fields rejected, the system achieved 99.8% character and 99.5% field success rate.

In the testing mode, the preprocessing module would separate characters if it can reliably do so, normalize them individually, and place them with gaps of ten blank pixels, in order to simplify the job of both the positioning and the classification modules. When it is impossible to segment individual characters, our system is still able to perform on the level of approximately 94% (since it has been trained on such data). The robustness of our system is an important factor in its success. Most other systems have substantial difficulties trying to recover from errors in segmentation.

## References

Benjio, Y., Le Cun, Y., and Henderson, D. (1994) Globally Trained Handwritten Word Recognizer Using Spatial Representation, Space Displacement Neural Networks and Hidden Markov Models. In Cowan, J.D., Tesauro, G., and Alspector, J. (eds.), *Advances in Neural Information Processing Systems 6*, pp. 937-944. San Mateo, CA: Morgan Kaufmann Publishers.

Gupta, A., Nagendraprasad, M.V., Liu, A., Wang, P.S.P., and Ayyadurai, S. (1993) An Integrated Architecture for Recognition of Totally Unconstrained Handwritten Numerals. *International Journal of Pattern Recognition and Artificial Intelligence* 7 (4), pp. 757-773.

Keeler, J. and Rumelhart, D.E. (1992) A Self-Organizing Integrated Segmentation and Recognition Neural Net. In Moody, J.E., Hanson, S.J., and Lippmann, R.P. (eds.), *Advances in Neural Information Processing Systems 4*, pp. 496-503. San Mateo, CA: Morgan Kaufmann Publishers.

Martin, G., Mosfeq, R., Chapman, D., and Pittman, J. (1993) Learning to See Where and What: Training a Net to Make Saccades and Recognize Handwritten Characters. In Hanson, S.J., Cowan, J.D., and Giles, C.L. (eds.), *Advances in Neural Information Processing Systems 5*, pp. 441-447. San Mateo, CA: Morgan Kaufmann Publishers.

Shustorovich, A. (1994) A Subspace Projection Approach to Feature Extraction: the Two-Dimensional Gabor Transform for Character Recognition. *Neural Networks* 7 (8), 1295-1301.

Shustorovich, A. and Thrasher, C.W. (1995) KODAK IMAGELINK™ OCR Numeric Handprint Module: Neural Network Positioning and Classification. Proceedings of Session 11 (Document Processing) of the industrial conference of *ICANN-95* Paris, October 9-13, 1995.

**Original Image with Detected Subimages**

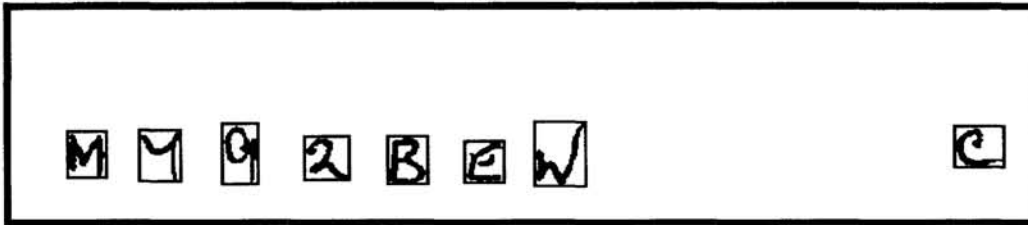

**Scaled Subimages**

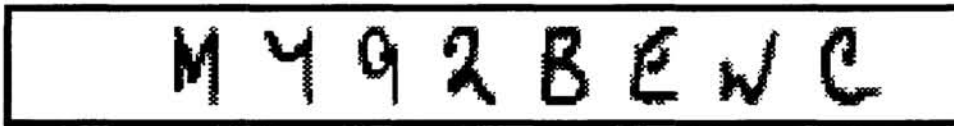

**Character Heart Index Waveform**

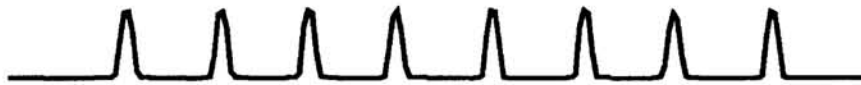

**Detected Character Hearts**

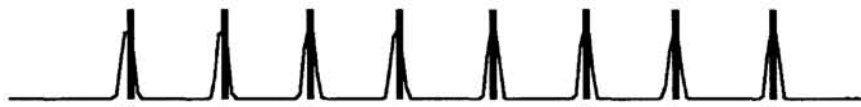

**Character Heart Images**

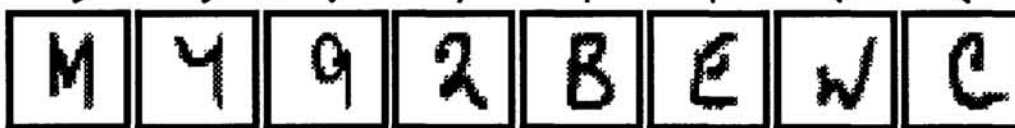

**Best Guess Characters**

MY9 Z B E WC

**Final Character String (After Post-Processing)**

M7 9 2 B E WC

Figure 1: An Example of a Field Processed by the System
Outline characters indicate low confidence.

Original Image with Detected Subimages

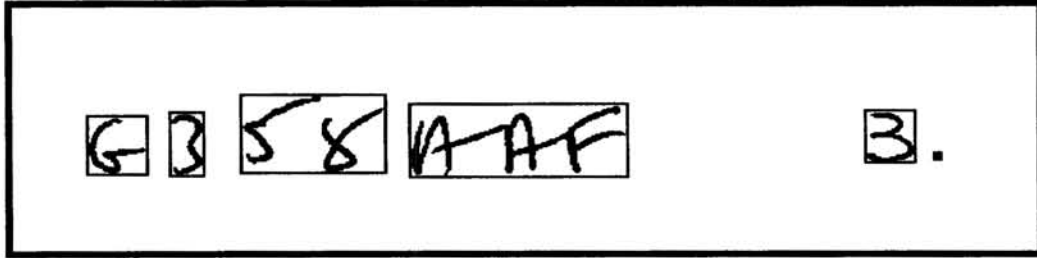

Scaled Subimages

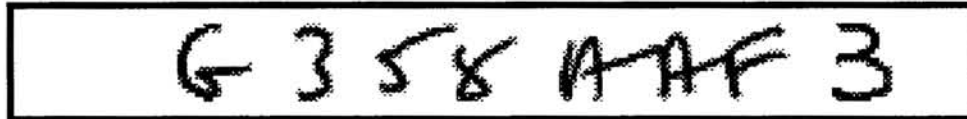

Character Heart Index Waveform

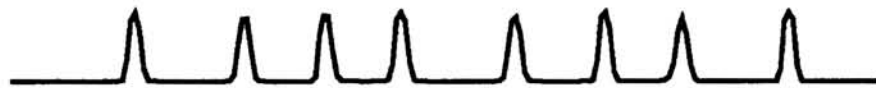

Detected Character Hearts

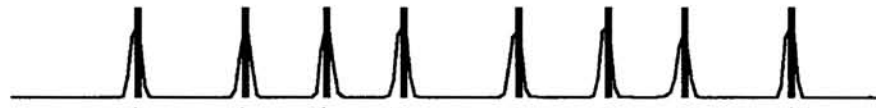

Character Heart Images

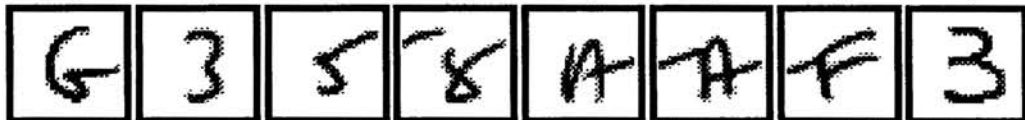

Best Guess Characters

# G 3 S 8 A A F 3

Final Character String (After Post-Processing)

# G 3 5 8 A A F 3

Figure 2: Another Example of a Field Processed by the System.